# Transelliptical Graphical Models

**Han Liu**
Department of Operations Research
and Financial Engineering
Princeton University, NJ 08544
hanliu@princeton.edu

**Fang Han**
Department of Biostatistics
Johns Hopkins University
Baltimore, MD 21210
fhan@jhsph.edu

**Cun-hui Zhang**
Department of Statistics
Rutgers University
Piscataway, NJ 08854
cunhui@stat.rutgers.edu

## Abstract

We advocate the use of a new distribution family—the transelliptical—for robust inference of high dimensional graphical models. The transelliptical family is an extension of the nonparanormal family proposed by Liu et al. (2009). Just as the nonparanormal extends the normal by transforming the variables using univariate functions, the transelliptical extends the elliptical family in the same way. We propose a nonparametric rank-based regularization estimator which achieves the parametric rates of convergence for both graph recovery and parameter estimation. Such a result suggests that the extra robustness and flexibility obtained by the semiparametric transelliptical modeling incurs almost no efficiency loss. We also discuss the relationship between this work with the transelliptical component analysis proposed by Han and Liu (2012).

## 1 Introduction

We consider the problem of learning high dimensional graphical models. In a typical setting, a $d$-dimensional random vector $X = (X_1, ..., X_d)^T$ can be represented as an undirected graph denoted by $G = (V, E)$, where $V$ contains nodes corresponding to the $d$ variables in $X$, and the edge set $E$ describes the conditional independence relationship among $X_1, ..., X_d$. Let $X_{\setminus\{i,j\}} := \{X_k : k \neq i, j\}$. We say the joint distribution of $X$ is Markov to $G$ if $X_i$ is independent of $X_j$ given $X_{\setminus\{i,j\}}$ for all $(i, j) \notin E$. While often $G$ is assumed given, here we want to estimate it from data.

Most graph estimation methods rely on the Gaussian graphical models, in which the random vector $X$ is assumed to be Gaussian: $X \sim N_d(\mu, \Sigma)$. Under this assumption, the graph $G$ is encoded by the precision matrix $\Theta := \Sigma^{-1}$. More specifically, no edge connects $X_j$ and $X_k$ if and only if $\Theta_{jk} = 0$. This problem of estimating $G$ is called *covariance selection* [5]. In low dimensions where $d < n$, [6, 7] develop a multiple testing procedure for identifying the sparsity pattern of the precision matrix. In high dimensions where $d \gg n$, [21] propose a neighborhood pursuit approach for estimating Gaussian graphical models by solving a collection of sparse regression problems using the Lasso [25, 3]. Such an approach can be viewed as a pseudo-likelihood approximation of the full likelihood. In contrast, [1, 30, 10] propose a penalized likelihood approach to directly estimate $\Omega$. [15, 14, 24] maximize the non-concave penalized likelihood to obtain an estimator with less bias than the traditional $L_1$-regularized estimator. Under the irrepresentable conditions [33, 31, 27], [22, 23] study the theoretical properties of the penalized likelihood methods. More recently, [29, 2] propose the graphical Dantzig selector and CLIME, which can be solved by linear programming and possess more favorable theoretical properties than the penalized likelihood approach.

Besides Gaussian models, [18] propose a semiparametric procedure named *nonparanormal* SKEP-TIC which extends the Gaussian family to the more flexible semiparametric Gaussian copula family. Instead of assuming $X$ follows a Gaussian distribution, they assume there exists a set of monotone functions $f_1, \ldots, f_d$, such that the transformed data $f(X) := (f_1(X_1), \ldots, f_d(X_d))^T$ is Gaussian. More details can be found in [18]. [32] has developed a scalable software package to implement these algorithms. In another line of research, [26] extends the Gaussian graphical models to the elliptical graphical models. However, for elliptical distributions, only the *generalized partial correlation graph* can be reliably estimated. These graphs only represent the conditional uncorrelatedness, but conditional independence, among variables. Therefore, by extending the Gaussian to the elliptical family, the gain in modeling flexibility is traded off with a loss in the strength of inference. In a related work, [9] provide a latent variable interpretation of the generalized partial correlation graph for multivariate $t$-distributions. An EM-type algorithm is proposed to fit the model for high dimensional data. However, the theoretical properties of their estimator is unknown.

In this paper, we introduce a new distribution family named *transelliptical graphical model*. A key concept is the *transelliptical distribution* [12]. The transelliptical distribution is a generalization of the nonparanormal distribution proposed by [18]. By mimicking how the nonparanormal extends the normal family, the transelliptical extends the elliptical family in the same way. The transelliptical family contains the nonparanomral family and elliptical family. To infer the graph structure, a rank-based procedure using the Kendall's tau statistic is proposed. We show such a procedure is adaptive over the transelliptical family: the procedure by default delivers a conditional uncorrelated graphs among certain latent variables; however, if the true distribution is the nonparanormal, the procedure automatically delivers the conditional independence graph. Computationally, the only extra cost is a one-pass data sort, which is almost negligible. Theoretically, even though the transelliptical family is much larger than the nonparanormal family, the same parametric rates of convergence for graph recovery and parameter estimation can be established. These results suggest that the transelliptical graphical model can be used routinely as a replacement of the nonparanormal models. Thorough numerical results are provided to back up our theory.

## 2 Background on Elliptical Distributions

Let $X$ and $Y$ be two random variables, we denote by $X \overset{d}{=} Y$ if they have the same distribution.

**Definition 2.1 (elliptical distribution [8]).** Let $\mu \in \mathbb{R}^d$ and $\Sigma \in \mathbb{R}^{d \times d}$ with $\text{rank}(\Sigma) = q \leq d$. A $d$-dimensional random vector $X$ has an elliptical distribution, denoted by $X \sim EC_d(\mu, \Sigma, \xi)$, if it has a stochastic representation: $X \overset{d}{=} \mu + \xi A U$, where $U$ is a random vector uniformly distributed on the unit sphere in $\mathbb{R}^q$, $\xi \geq 0$ is a scalar random variable independent of $U$, $A \in \mathbb{R}^{d \times q}$ is a deterministic matrix such that $AA^T = \Sigma$.

**Remark 2.1.** An equivalent definition of an elliptical distribution is that its characteristic function can be written as $\exp(it^T \mu)\phi(t^T \Sigma t)$, where $\phi$ is a properly-defined characteristic function which has a one-to-one mapping with $\xi$ in Definition 2.1. In this setting we denote by $X \sim EC_d(\mu, \Sigma, \phi)$.

An elliptical distribution does not necessarily have a density. One example is the rank-deficient Gaussian. More examples can be found in [11]. However, when the random variable $\xi$ is absolutely continuous with respect to the Lebesgue measure and $\Sigma$ is non-singular, the density of $X$ exists and has the form

$$p(x) = |\Sigma|^{-1/2} g \left( (x - \mu)^T \Sigma^{-1} (x - \mu) \right), \tag{1}$$

where $g(\cdot)$ is a scale function uniquely determined by the distribution of $\xi$. In this case, we can also denote it as $X \sim EC_d(\mu, \Sigma, g)$. Many multivariate distributions belong to the elliptical family. For example, when $g(x) = (2\pi)^{-d/2} \exp\{-x/2\}$, $X$ is $d$-dimensional Gaussian. Another important subclass is the multivariate $t$-distribution with the degrees of freedom $v$, in which, we choose

$$g(x) = c_v \frac{\Gamma\left(\frac{v+d}{2}\right)}{(v\pi)^{\frac{d}{2}}\Gamma(\frac{v}{2})} \left(1 - \frac{c_v^2 x}{v}\right)^{-\frac{v+d}{2}}, \tag{2}$$

where $c_v$ is a normalizing constant.

The model family in Definition 2.1 is not identifiable. For example, given $X \sim EC_d(\mu, \Sigma, \xi)$ with $\text{rank}(\Sigma) = q$, there will be multiple $A$s corresponding to the same $\Sigma$. i.e., there exist $A_1 \neq A_2 \in$

$\mathbb{R}^{d \times q}$ such that $A_1 A_1^T = A_2 A_2^T = \Sigma$. For some constant $c \neq 0$, we define $\xi^* = \xi/c$ and $A^* = c \cdot A$, then $\xi A U = \xi^* A^* U$. Therefore, the matrix $\Sigma$ is unique only up to a constant scaling. To make the model identifiable, we impose the condition that $\max\{\text{diag}(\Sigma)\} = 1$. More discussions about the identifiability issue can be found in [12].

## 3 Transelliptical Graphical Models

In this paper we only consider distributions with continuous marginals. We introduce the transelliptical graphical models in analogy to the nonparanormal graphical models [19, 18]. The key concept is transelliptical distribution which is also introduced in [12]. However, the definition of transelliptical distribution in this paper is slightly more restrictive than that in [12] due to the complication of graphical modeling. More specifically, let

$$\mathcal{R}_d^+ := \{\Sigma \in \mathbb{R}^{d \times d} : \Sigma^T = \Sigma, \text{diag}(\Sigma) = \mathbf{1}, \Sigma \succ \mathbf{0}\}, \tag{3}$$

we define the transelliptical distribution as follows:

**Definition 3.1 (transelliptical distribution).** A continuous random vector $X = (X_1, \ldots, X_d)^T$ is transelliptical, denoted by $X \sim TE_d(\Sigma, \xi; f_1, \ldots, f_d)$, if there exists a set of monotone univariate functions $f_1, \ldots, f_d$ and a nonnegative random variable $\xi$ satisfying $\mathbb{P}(\xi = 0) = 0$, such that

$$(f_1(X_1), \ldots, f_d(X_d))^T \sim EC_d(\mathbf{0}, \Sigma, \xi), \quad \text{where } \Sigma \in \mathcal{R}_d^+. \tag{4}$$

Here, $\Sigma$ is called *latent generalized correlation matrix*[1].

We then discuss the relationship between the transelliptical family with the nonparanormal family, which is defined as follows:

**Definition 3.2 (nonparanormal distribution).** A ramdom vector $X = (X_1, \ldots, X_d)^T$ is nonparanormal, denoted by $X \sim NPN_d(\Sigma; f_1, \ldots, f_d)$, if there exist monotone functions $f_1, \ldots, f_d$ such that $(f_1(X_1), \ldots, f_d(X_d))^T \sim N_d(\mathbf{0}, \Sigma)$, where $\Sigma \in \mathcal{R}_d^+$ is called *latent correlation matrix*.

From Definitions 3.1 and 3.2, we see the transelliptical is a strict extension of the nonparanormal. Both families assume there exits a set of univariate transformations such that the transformed data follow a base distribution: the nonparanormal exploits a normal base distribution; while the transelliptical exploits an elliptical base distribution. In the nonparanormal, $\Sigma$ is the correlation matrix for the latent normal, therefore it is called latent correlation matrix; In the transelliptical, $\Sigma$ is the generalized correlation matrix for the latent elliptical distribution, therefore it is called latent generalized correlation matrix.

We now define the transelliptical graphical models. Let $X \sim TE_d(\Sigma, \xi; f_1, \ldots, f_d)$ where $\Sigma \in \mathcal{R}_d^+$ is the latent generalized correlation matrix. In this paper, we always assume the second moment $\mathbb{E}\xi^2 < \infty$. We define $\Theta := \Sigma^{-1}$ to be the *latent generalized concentration matrix*. Let $\Theta_{jk}$ be the element of $\Theta$ on the $j$-th row and $k$-th column. We define the *latent generalized partial correlation matrix* $\Gamma$ as $\Gamma_{jk} := -\Theta_{jk}/\sqrt{\Theta_{jj} \cdot \Theta_{kk}}$. Let $\text{diag}(A)$ be the matrix $A$ with off-diagonal elements replaced by zero and $A^{1/2}$ be the squared root matrix of $A$. It is easy to see that

$$\Gamma = -[\text{diag}(\Sigma^{-1})]^{-1/2} \Sigma^{-1} [\text{diag}(\Sigma^{-1})]^{-1/2}. \tag{5}$$

Therefore, $\Gamma$ has the same nonzero pattern as $\Sigma^{-1}$. We then define a undirected graph $G = (V, E)$: the vertex set $V$ contains nodes corresponding to the $d$ variables in $X$, and the edge set $E$ satisfies

$$(X_j, X_k) \in E \text{ if and only if } \Gamma_{jk} \neq 0 \text{ for } j, k = 1, \ldots, d. \tag{6}$$

Given a graph $G$, we define $\mathcal{R}_d^+(G)$ to be the set containing all the $\Sigma \in \mathcal{R}_d^+$ with zero entries at the positions specified by the graph $G$. The transelliptical graphical model induced by $G$ is defined as:

**Definition 3.3 (transelliptical graphical model).** The transelliptical graphical model induced by a graph $G$, denoted by $\mathcal{P}(G)$, is defined to be the set of distributions:

$$\mathcal{P}(G) := \left\{ \text{all the transelliptical distributions } TE_d(\Sigma, \xi; f_1, \ldots, f_d) \text{ satisfying } \Sigma \in \mathcal{R}_d^+(G) \right\}. \tag{7}$$

In the rest of this section, we prove some properties of the transelliptical family and discuss the interpretation of the meaning of the graph $G$. This graph is called *latent generalized partial correlation graph*. First, we show the transelliptical family is closed under marginalization and conditioning.

**Lemma 3.1.** *Let* $X := (X_1, \ldots, X_d)^T \sim TE_d(\Sigma, \xi; f_1, \ldots, f_d)$. *The marginal and the conditional distributions of* $(X_1, X_2)^T$ *given the remaining variables are still transellpitical.*

*Proof.* Since $X \sim TE_d(\Sigma, \xi; f_1, \ldots, f_d)$, we have $(f_1(X_1), \ldots, f_d(X_d))^T \sim EC_d(\mathbf{0}, \Sigma, \xi)$. Let $Z_j := f_j(X_j)$ for $j = 1, \ldots, d$. From Theorem 2.18 of [8], the marginal distribution of $(Z_1, Z_2)^T$ and the conditional distribution of $(Z_1, Z_2)^T$ given the remaining $Z_3, \ldots, Z_d$ are both elliptical. By definition, the marginal distribution of $(X_1, X_2)^T$ is transelliptical. To see the conditional case, since $X$ has continuous marginals and $f_1, \ldots, f_d$ are monotone, the distribution of $(X_1, X_2)^T$ conditional on $X_{\backslash\{1,2\}}$ is the same as conditional on $Z_{\backslash\{1,2\}}$. Combined with the fact that $Z_1 = f_1(X_1)$, $Z_2 = f_2(X_2)$, we know that $(X_1, X_2)^T \mid X_{\backslash\{1,2\}}$ follows a transelliptical distribution. $\qquad\square$

From (5), we see the matrices $\Gamma$ and $\Theta$ have the same nonzero pattern, therefore, they encode the same graph $G$. Let $X \sim TE_d(\Sigma, \xi; f_1, \ldots, f_d)$. The next lemma shows that, if the second moment of $X$ exists, the absence of an edge in the graph $G$ is equivalent to the pairwise conditional uncorrelatedness of two corresponding latent variables.

**Lemma 3.2.** *Let* $X := (X_1, \ldots, X_d)^T \sim TE_d(\Sigma, \xi; f_1, \ldots, f_d)$ *with* $\mathbb{E}\xi^2 < \infty$, *and* $Z_j := f_j(X_j)$ *for* $j = 1, \ldots, d$. $\Gamma_{jk} = 0$ *if and only if* $Z_j$ *and* $Z_k$ *are conditionally uncorrelated given* $Z_{\backslash\{j,k\}}$.

*Proof.* Let $Z := (Z_1, \ldots, Z_d)^T$. Since $X \sim TE_d(\Sigma, \xi; f_1, \ldots, f_d)$, we have $Z \sim EC_d(\mathbf{0}, \Sigma, \xi)$. Therefore, the latent generalized correlation matrix $\Sigma$ is the generalized correlation matrix of the latent variable $Z$. It suffices to prove that, for elliptical distributions with $\mathbb{E}\xi^2 < \infty$, the generalized partial correlation matrix $\Gamma$ as defined in (5) encodes the conditional uncorrelatedness among the variables. Such a result has been proved in the section 2 of [26]. $\qquad\square$

Let $A, B, C \subset \{1, \ldots, d\}$. We say $C$ separates $A$ and $B$ in the graph $G$ if any path from a node in $A$ to a node in $B$ goes through at least one node in $C$. We denote by $X_A$ the subvector of $X$ indexed by $A$. The next lemma implies the equivalence between the pairwise and global conditional uncorrelatedness of the latent variables for the transelliptical graphical models. This lemma connects the graph theory with probability theory.

**Lemma 3.3.** *Let* $X \sim TE_d(\Sigma, \xi; f_1, \ldots, f_d)$ *be any element of the transelliptical graphical model* $\mathcal{P}(G)$ *satisfying* $\mathbb{E}\xi^2 < \infty$. *Let* $Z := (Z_1, \ldots, Z_d)^T$ *with* $Z_j = f_j(X_j)$ *and* $A, B, C \subset \{1, \ldots, d\}$. *Then* $C$ *separates* $A$ *and* $B$ *in* $G$ *if and only if* $Z_A$ *and* $Z_B$ *are conditional uncorrelated given* $Z_C$.

*Proof.* By definition, we know $Z \sim EC_d(\mathbf{0}, \Sigma, \xi)$. It then suffices to show the pairwise conditional uncorrelatedness implies the global conditional uncorrelatedness for the elliptical family. This follows from the same induction argument as in Theorem 3.7 of [16]. $\qquad\square$

Compared with the nonparanormal graphical model, the transelliptical graphical model gains a lot on modeling flexibility, but at the price of inferring a weaker notion of graphs: a missing edge in the graph only represents the conditional uncorrelatedness of the latent variables. The next lemma shows that we do not lose any thing compared with the nonparanormal graphical model. The proof of this lemma is simple and is omitted. Some related discussions can be found in [19].

**Lemma 3.4.** *Let* $X \sim TE_d(\Sigma, \xi; f_1, \ldots, f_d)$ *be a member of the transelliptical graphical model* $\mathcal{P}(G)$. *If* $X$ *is also nonparanormal, the graph* $G$ *encodes the conditional independence relationship of* $X$ *(In other words, the distribution of* $X$ *is Markov to* $G$).

## 4 Rank-based Regularization Estimator

In this section, we propose a nonparametric rank-based regularization estimator which achieves the optimal parametric rates of convergence for both graph recovery and parameter estimation. The main idea of our procedure is to treat the marginal transformation functions $f_j$ and the generating variable $\xi$ as nuisance parameters, and exploit the nonparametric Kendall's tau statistic to directly estimate the latent generalized correlation matrix $\Sigma$. The obtained correlation matrix estimate is then plugged into the CLIME procedure to estimate the sparse latent generalized concentration matrix $\Theta$. From the previous discussion, we know the graph $G$ is encoded by the nonzero pattern of $\Theta$. We then get a graph estimator by thresholding the estimated $\widehat{\Theta}$.

## 4.1 The Kendall's tau Statistic and its Invariance Property

Let $x^1, \ldots, x^n \in \mathbb{R}^d$ be $n$ observations of a random vector $X \sim TE_d(\Sigma, \xi; f_1, \ldots, f_d)$. Our task is to estimate the latent generalized concentration matrix $\Theta := \Sigma^{-1}$. The Kendall's tau is defined as:

$$\widehat{\tau}_{jk} = \frac{2}{n(n-1)} \sum_{1 \le i < i' \le n} \text{sign}\left(x_j^i - x_j^{i'}\right)\left(x_k^i - x_k^{i'}\right), \tag{8}$$

which is a monotone transformation-invariant correlation between the empirical realizations of two random variables $X_j$ and $X_k$. Let $\widetilde{X}_j$ and $\widetilde{X}_k$ be two independent copies of $X_j$ and $X_k$. The population version of the Kendall's tau statistic is $\tau_{jk} := \text{Corr}\big(\text{sign}(X_j - \widetilde{X}_j), \text{sign}(X_k - \widetilde{X}_k)\big)$.

Let $X \sim TE_d(\Sigma, \xi; f_1, \ldots, f_d)$, the following theorem from [12] illustrates an important relationship between the population Kendall's tau statistic $\tau_{jk}$ and the latent generalized correlation coefficient $\Sigma_{jk}$.

**Theorem 4.1 (Invariance Property of Kendall's tau Statistic[12]).** *Let $X := (X_1, \ldots, X_d)^T \sim TE_d(\Sigma, \xi; f_1, \ldots, f_d)$. We denote $\tau_{jk}$ to be the population Kendall's tau statistic between $X_j$ and $X_k$. Then $\Sigma_{jk} = \sin\left(\frac{\pi}{2}\tau_{jk}\right)$.*

## 4.2 Rank-based Regularization Method

We start with some notations. We denote by $I(\cdot)$ to be the indicator function and $\boldsymbol{I}_d$ be the identity matrix. Given a matrix $A$, we define $\|A\|_{\max} := \max_{jk}|A_{jk}|$ and $\|A\|_1 := \sum_{jk}|A_{jk}|$.

Motivated by Theorem 4.1, we define $\widehat{S} = [\widehat{S}_{jk}] \in \mathbb{R}^{d \times d}$ to estimate $\Sigma$:

$$\widehat{S}_{jk} = \sin\left(\frac{\pi}{2}\widehat{\tau}_{jk}\right) \cdot I(j \ne k) + I(j = k). \tag{9}$$

We then plug $\widehat{S}$ into the CLIME estimator [2] to get the final parameter and graph estimates. More specifically, the latent generalized concentration matrix $\Theta$ can be estimated by solving

$$\widehat{\Theta} = \arg\min_{\Theta} \sum_{j,k} |\Theta_{jk}| \text{ s.t. } \|\widehat{S}\Theta - \boldsymbol{I}_d\|_{\max} \le \lambda, \tag{10}$$

where $\lambda > 0$ is a tuning parameter. [2] show that this optimization can be decomposed into $d$ vector minimization problems, each of which can be reformulated as a linear program. Thus it has the potential to scale to very large problems. Once $\widehat{\Theta}$ is obtained, we can apply an additional thresholding step to estimate the graph $G$. For this, we define a graph estimator $\widehat{G} = (V, \widehat{E})$, in which an edge $(j, k) \in \widehat{E}$ if $\widehat{\Theta}_{jk} \ge \gamma$. Here $\gamma$ is another tuning parameter.

Compared with the original CLIME, the extra cost of our rank-based procedure is the computation of $\widehat{S}$, which requires us to evaluate $d(d-1)/2$ pairwise Kendal's tau statistics. A naive implementation of the Kendall's tau requires $O(n^2)$ computation. However, efficient algorithm based on sorting and balanced binary trees has been developed to calculate the Kendall's tau statistic with a computational complexity $O(n \log n)$ [4]. Therefore, the incurred computational burden is negligible.

**Remark 4.1.** Similar rank-based procedures have been discussed in [19, 18, 28]. Unlike our work, they focus on the more restrictive nonparanromal family and discuss several rank-based procedures using the normal-score, Spearman's rho, and Kendall's tau. Unlike our results, they advocate the use of the Spearman's rho and normal-score correlation coefficients. Their main concern is that, within the more restrictive nonparanormal family, the Spearman's rho and normal-score correlations are slightly easier to compute and have smaller asymptotic variance. In contrast to their results, the new insight obtained from this current paper is that we advocate the usage of the Kendall's tau due to its invariance property within the much larger transelliptical family. In fact, we can show that the Spearman's rho is not invariant within the transelliptical family unless the true distribution is nonparanormal. More details on this issue can be found in [8].

## 5 Asymptotic Properties

We analyze the theoretical properties of the rank-based regularization estimator proposed in Section 4.2. Our main result shows: under the same conditions on $\Sigma$ that ensure the parameter estimation

and graph recovery consistency of the original CLIME estimator for Gaussian graphical models, our rank-based regularization procedure achieves exactly the same parametric rates of convergence for both parameter estimation and graph recovery for the much larger transelliptical family. This result suggests that the transelliptical graphical model can be used as a safe replacement of the Gaussian graphical models, the nonparanormal graphical models, and the elliptical graphical models.

We introduce some additional notations. Given a symmetric matrix $A$, for $0 \leq q < 1$, we define $\|A\|_{L_q} := \max_i \sum_j |A_{ij}|^q$ and the spectral norm $\|A\|_{L_2}$ to be its largest eigenvalue. We define

$$\mathcal{S}_d(q, s, M) := \{\Theta : \|\Theta\|_{L_1} \leq M \text{ and } \|\Theta\|_{L_q} \leq s\}. \tag{11}$$

For $q = 0$, the class $\mathcal{S}_d(0, s, M)$ contains all the $s$-sparse matrices. Our main result is Theorem 5.1

**Theorem 5.1.** *Let* $X \sim TE_d(\Sigma, \xi; f_1, \ldots, f_d)$ *with* $\Sigma \in \mathcal{R}_d^+$ *and* $\Theta := \Sigma^{-1} \in \mathcal{S}_d(q, S, M)$ *with* $0 \leq q < 1$. *Let* $\widehat{\Theta}$ *be defined in* (10). *There exist constants* $C_0$ *and* $C_1$ *only depending on* $q$, *such that, whenever* $\lambda = C_0 M \sqrt{(\log d)/n}$, *with probability no less than* $1 - d^{-2}$, *we have*

$$(\text{Parameter estimation}) \qquad \|\widehat{\Theta} - \Theta\|_{L_2} \leq C_1 M^{2-2q} \cdot s \cdot \left( \frac{\log d}{n} \right)^{(1-q)/2}. \tag{12}$$

*Let* $\widehat{G}$ *be the graph estimator defined in Section 4.2 with the additional tuning parameter* $\gamma = 4M\lambda$. *If we further assume* $\Theta \in \mathcal{S}_d(0, s, M)$ *and* $\min_{j,k:|\Theta_{jk}|\neq 0} |\Theta_{jk}| \geq 2\gamma$, *then*

$$(\text{Graph recovery}) \qquad \mathbb{P}\left(\widehat{G} \neq G\right) \geq 1 - o(1), \tag{13}$$

*where* $G$ *is the graph determined by the nonzero pattern of* $\Theta$.

*Proof.* The difference between the rank-based CLIME and the original CLIME is that we replace the Pearson correlation coefficient matrix $\widehat{R}$ by the Kendall's tau matrix $\widehat{S}$. By examing the proofs of Theorems 1 and 7 in [2], the only property needed of $\widehat{R}$ is an exponential concentration inequality

$$\mathbb{P}\left(|\widehat{R}_{jk} - \Sigma_{jk}| > t\right) \leq c_1 \exp(-c_2 n t^2)$$

. Therefore, it suffices if we can prove a similar concentration inequality for $|\widehat{S}_{jk} - \Sigma_{jk}|$. Since

$$\widehat{S} = \sin\left(\frac{\pi}{2}\widehat{\tau}_{jk}\right) \quad \text{and} \quad \Sigma_{jk} = \sin\left(\frac{\pi}{2}\tau_{jk}\right),$$

we have $|\widehat{S}_{jk} - \Sigma_{jk}| \leq |\widehat{\tau}_{jk} - \tau|$. Therefore, we only need to prove

$$\mathbb{P}\left(|\widehat{\tau}_{jk} - \tau_{jk}| > t\right) \leq \exp\left(-n t^2/(2\pi)\right).$$

This result holds since $\widehat{\tau}_{jk}$ is a U-statistic: $\widehat{\tau}_{jk} = \frac{2}{n(n-1)} \sum_{1 \leq i < i' \leq n} K_\tau(x^i, x^{i'})$, where $K_\tau(x^i, x^{i'}) = \text{sign}(x_j^i - x_j^{i'})(x_k^i - x_k^{i'})$ is a bounded kernel between -1 and 1. The result follows from the Hoeffding's inequality for U-statistic [13]. $\square$

# 6 Numerical Experiments

We investigate the empirical performance of the rank-based regularization estimator. We compare it with the following methods: (1) Pearson: the CLIME using the Pearson sample correlation; (2) Kendall: the CLIME using the Kendall's tau; (3) Spearman: the CLIME using the Spearman's rho; (4) NPN: the CLIME using the original nonparanormal correlation estimator [19]; (5) NS: the CLIME using the normal score correlation. The later three methods are discussed under the nonparanormal graphical model and we refer to [18] for detailed descriptions.

## 6.1 Simulation Studies

We adopt the same data generating procedure as in [18]. To generate a $d$-dimensional sparse graph $G = (V, E)$ where $V = \{1, \ldots, d\}$ correspond to variables $X = (X_1, \ldots, X_d)$, we associate each index $j \in \{1, \ldots, d\}$ with a bivariate data point $(Y_j^{(1)}, Y_j^{(2)}) \in [0,1]^2$ where $Y_1^{(k)}, \ldots, Y_n^{(k)} \sim$ Uniform$[0, 1]$ for $k = 1, 2$. Each pair of vertices $(i, j)$ is included in the edge set $E$ with probability $\mathbb{P}((i, j) \in E) = \exp(-\|y_i - y_j\|_n^2/0.25)/\sqrt{2\pi}$, where $y_i := (y_i^{(1)}, y_i^{(2)})$ is the empirical observation of $(Y_i^{(1)}, Y_i^{(2)})$ and $\|\cdot\|_n$ represents the Euclidean distance. We restrict the maximum degree of the

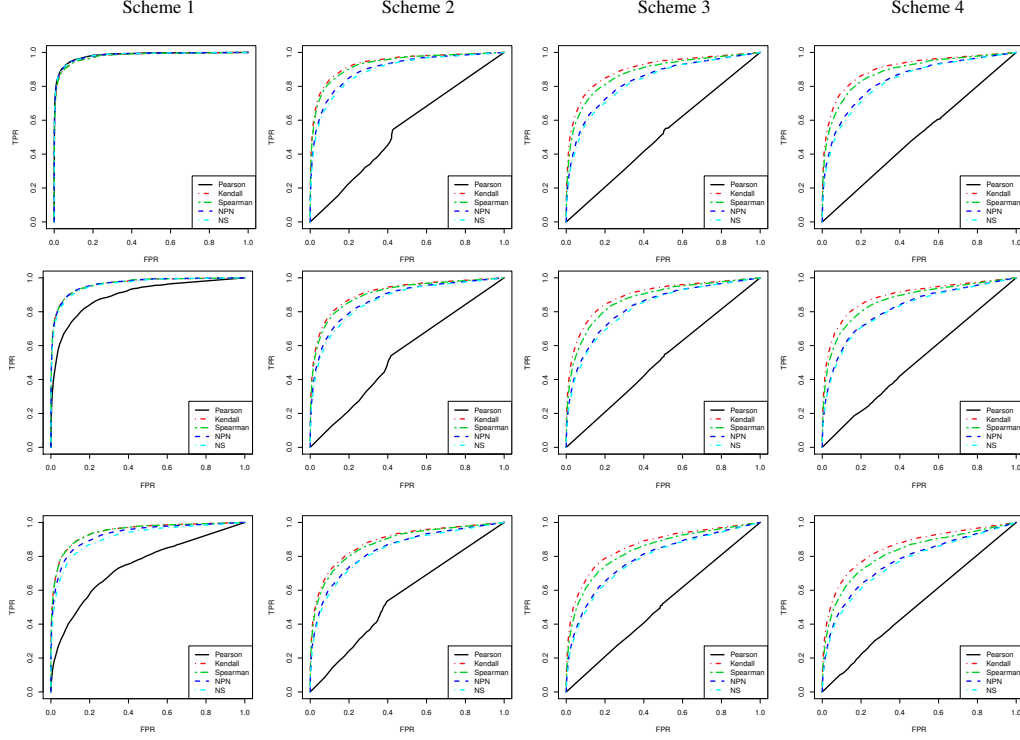

Figure 1: ROC curves for different methods in generating schemes 1 to 4 and different contamination level $r = 0, 0.02, 0.05$ (top, middle, bottom) using the CLIME. Here $n = 400$ and $d = 100$.

graph to be 4 and build the inverse correlation matrix $\Omega$ according to $\Omega_{jk} = 1$ if $j = k$, $\Omega_{jk} = 0.145$ if $(j, k) \in E$, and $\Omega_{jk} = 0$ otherwise. The value $0.145$ guarantees the positive definiteness of $\Omega$. Let $\Sigma = \Omega^{-1}$. To obtain the correlation matrix, we rescale $\Sigma$ so that all its diagonal elements are 1.

In the simulated study we randomly sample $n$ data points from a certain transelliptical distribution $X \sim TE_d(\Sigma, \xi; f_1, \ldots, f_d)$. We set $d = 100$. To determine the transelliptical distribution, we first generate $\Sigma$ as discussed in the previous paragraph. Secondly, three types of $\xi$ are considered:

(1) $\xi^{(1)} \sim \chi_d$, i.e., $\xi$ follows a chi-distribution with degree of freedom $d$;

(2) $\xi^{(2)} \stackrel{d}{=} \xi_1^*/\xi_2^*$, $\xi_1^* \sim \chi_d$, $\xi_2^* \sim \chi_1$, $\xi_1^*$ is independent of $\xi_2^*$;

(3) $\xi^{(3)} \sim F(d, 1)$, i.e., $\xi$ follows an $F$-distribution with degree of freedom $d$ and 1.

Thirdly, two type of transformation functions $f = \{f_j\}_{j=1}^d$ are considered:

(1) **linear transformation:** $f^{(1)} = \{f_0, \ldots, f_0\}$ with $f_0(x) = x$;

(2) **nonlinear transformation:** $f^{(2)} = \{f_1, \ldots, f_d\} = \{h_1, h_2, h_3, h_4, h_5, h_1, h_2, h_3, h_4, h_5, \ldots\}$, where $h_1^{-1}(x) := x$, $h_2^{-1}(x) := \frac{\text{sign}(x)|x|^{1/2}}{\sqrt{\int |t|\phi(t)dt}}$, $h_3^{-1}(x) := \frac{x^3}{\sqrt{\int t^6\phi(t)dt}}$, $h_4^{-1}(x) := \frac{\Phi(x) - \int \Phi(t)\phi(t)dt}{\sqrt{\int (\Phi(y) - \int \Phi(t)\phi(t)dt)^2 \phi(y)dy}}$, $h_5^{-1}(x) := \frac{\exp(x) - \int \exp(t)\phi(t)dt}{\sqrt{\int (\exp(y) - \int \exp(t)\phi(t)dt)^2 \phi(y)dy}}$.

We consider the following four data generating schemes:

- **Scheme 1:** $X \sim TE_d(\Sigma, \xi^{(1)}; f^{(1)})$, i.e., $X \sim N(0, \Sigma)$.

- **Scheme 2:** $X \sim TE_d(\Sigma, \xi^{(2)}; f^{(1)})$, i.e., $X$ follows the multivariate Cauchy.

- **Scheme 3:** $X \sim TE_d(\Sigma, \xi^{(3)}; f^{(1)})$, i.e., the distribution is highly related to the multivariate t.

- **Scheme 4:** $X \sim TE_d(\Sigma, \xi^{(3)}; f^{(2)})$.

To evaluate the robustness of different methods, let $r \in [0, 1)$ represent the proportion of samples being contaminated. For each dimension, we randomly select $\lfloor nr \rfloor$ entries and replace them with

either 5 or -5 with equal probability. The final data matrix we obtained is $\boldsymbol{X} \in \mathbb{R}^{n \times d}$. Here we pick $r = 0, 0.02$ or $0.05$. Under the Scheme 1 to Scheme 4 with different levels of contamination ($r = 0, 0.02$ or $0.05$), we repeatedly generate the data matrix $\boldsymbol{X}$ for 100 times and compute the averaged False Positive Rates and False Negative Rates using a path of tuning parameters $\lambda$ from $0.01$ to $0.5$ and $\gamma = 10^{-5}$. The feature selection performances of different methods are evaluated by plotting $(\mathrm{FPR}(\lambda), 1 - \mathrm{FNR}(\lambda))$. The corresponding ROC curves are presented in Figure 1. We see: (1) when the data are perfectly Gaussian without contamination, all methods perform well; (2) when data are non-Gaussian, with outliers existing or latent elliptical distribution different from the Gaussian, Kendall is better than the other methods in terms of achieving a lower $\mathrm{FPR} + \mathrm{FNR}$.

## 6.2 Equities Data

We compare different methods on the stock price data from Yahoo! Finance (`finance.yahoo.com`). We collect the daily closing prices for 452 stocks that are consistently in the S&P 500 index between January 1, 2003 through January 1, 2008. This gives us altogether 1,257 data points, each data point corresponding to the vector of closing prices on a trading day. With $S_{t,j}$ denoting the closing price of stock $j$ on day $t$, we consider the variables $X_{tj} = \log(S_{t,j}/S_{t-1,j})$ and build graphs over the indices $j$. Though a time series, we treat the instances $X_t$ as independent replicates.

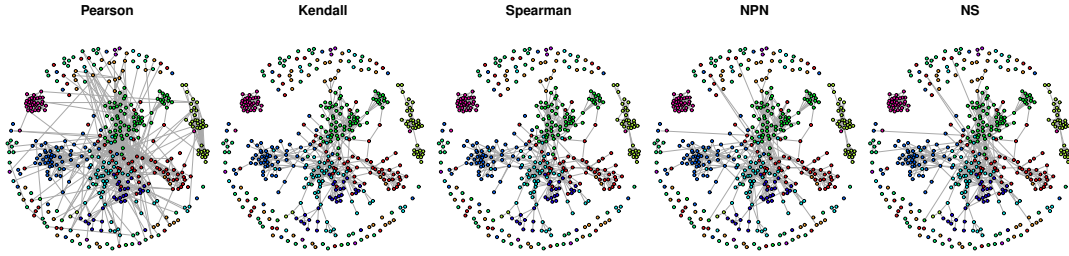

Figure 2: The graph estimated from the S&P 500 stock data from Jan. 1, 2003 to Jan. 1, 2008 using Pearson, Kendall,Spearman, NPN, NS (left to right). The nodes are colored according to their GICS sector categories.

The 452 stocks are categorized into 10 Global Industry Classification Standard (GICS) sectors, including `Consumer Discretionary` (70 stocks), `Consumer Staples` (35 stocks), `Energy` (37 stocks), `Financials` (74 stocks), `Health Care` (46 stocks), `Industrials` (59 stocks), `Information Technology` (64 stocks) `Telecommunications Services` (6 stocks), , `Materials` (29 stocks), and `Utilities` (32 stocks).

Figure 2 illustrates the estimated graphs using the same layout, the nodes are colored according to the GICS sector of the corresponding stock. The tuning parameter is automatically selected using a stability based approach named StARS [20]. We see that different methods get slightly different graphs. The layout is drawn by a force-based algorithm using the estimated graph from the Kendall. We see the stocks from the same GICS sector tends to be grouped with each other, suggesting that our method delivers an informative graph estimate.

## 7 Discussion and Comparison with Related Work

The transelliptical distribution is also proposed by [12] for semiparametric scale-invariant principle component analysis. Though both papers are based on the transelliptical family, the core idea and analysis are fundamentally different. For scale-invariant principle component analysis, we impose structural assumption of the latent generalized correlation matrix; For graph estimation, we impose structural assumption on the latent generalized concentration matrix. Since the latent generalized correlation matrix encodes marginal uncorrelatedness while the latent generalized concentration matrix encodes conditional uncorrelatedness of the variables, the analysis of the population models are orthogonal and complementary to each other. In particular, for graphical models, we need to characterize the properties of marginal and conditional distributions of a transelliptical distribution. These properties are not needed for principle component analysis. Moreover, the model interpretation of the inferred transelliptical graph is very nontrivial. In a longer version technical report [17], we provide a three-layer hierarchal interpretation of the estimated transelliptical graphical model and sharply characterize the relationships between nonparnaormal, elliptical, meta-elliptical, and transelliptical families. This research was supported by NSF award IIS-1116730.

## Footnotes

[1]One thing to note is that in [12], the condition that $\Sigma \in \mathcal{R}_{d+}$ is not required.

# References

[1] O. Banerjee, L. E. Ghaoui, and A. d'Aspremont. Model selection through sparse maximum likelihood estimation. *Journal of Machine Learning Research*, 9(3):485–516, 2008.

[2] T. Cai, W. Liu, and X. Luo. A constrained $\ell_1$ minimization approach to sparse precision matrix estimation. *Journal of the American Statistical Association*, 106(494):594–607, 2011.

[3] S. Chen, D. Donoho, and M. Saunders. Atomic decomposition by basis pursuit. *SIAM Journal on Scientific Computing*, 20(1):33–61, 1998.

[4] David Christensen. Fast algorithms for the calculation of Kendall's $\tau$. *Computational Statistics*, 20(1):51–62, 2005.

[5] A. Dempster. Covariance selection. *Biometrics*, 28:157–175, 1972.

[6] M. Drton and M. Perlman. Multiple testing and error control in Gaussian graphical model selection. *Statistical Science*, 22(3):430–449, 2007.

[7] M. Drton and M. Perlman. A SINful approach to Gaussian graphical model selection. *Journal of Statistical Planning and Inference*, 138(4):1179–1200, 2008.

[8] KT Fang, S. Kotz, and KW Ng. Symmetric multivariate and related distributions. *Chapman&Hall, London*, 1990.

[9] Michael A. Finegold and Mathias Drton. Robust graphical modeling with t-distributions. In *Proceedings of the Twenty-Fifth Conference on Uncertainty in Artificial Intelligence*, UAI '09, pages 169–176, 2009.

[10] J. Friedman, T. Hastie, and R. Tibshirani. Sparse inverse covariance estimation with the graphical lasso. *Biostatistics*, 9(3):432–441, 2008.

[11] P.R. Halmos. *Measure theory*, volume 18. Springer, 1974.

[12] F. Han and H. Liu. Tca: Transelliptical principal component analysis for high dimensional non-gaussian data. *Technical Report*, 2012.

[13] Wassily Hoeffding. Probability Inequalities for Sums of Bounded Random Variables. *Journal of the American Statistical Association*, 58(301):13–30, 1963.

[14] A. Jalali, C. Johnson, and P. Ravikumar. High-dimensional sparse inverse covariance estimation using greedy methods. *International Conference on Artificial Intelligence and Statistics*, 2012. to appear.

[15] C. Lam and J. Fan. Sparsistency and rates of convergence in large covariance matrix estimation. *Annals of Statistics*, 37:42–54, 2009.

[16] Steffen L. Lauritzen. *Graphical Models*. Oxford University Press, 1996.

[17] H. Liu, F. Han, and Zhang C-H. Transelliptical graphical modeling under a hierarchical latent variable framework. *Technical Report*, 2012.

[18] H. Liu, F. Han, M. Yuan, J. Lafferty, and L. Wasserman. High dimensional semiparametric gaussian copula graphical models. *Annals of Statistics*, 2012.

[19] H. Liu, J. Lafferty, and L. Wasserman. The nonparanormal: Semiparametric estimation of high dimensional undirected graphs. *Journal of Machine Learning Research*, 10:2295–2328, 2009.

[20] Han Liu, Kathryn Roeder, and Larry Wasserman. Stability approach to regularization selection (stars) for high dimensional graphical models. In *Proceedings of the Twenty-Third Annual Conference on Neural Information Processing Systems (NIPS)*, 2010.

[21] N. Meinshausen and P. Bühlmann. High dimensional graphs and variable selection with the lasso. *Annals of Statistics*, 34(3):1436–1462, 2006.

[22] P. Ravikumar, M. Wainwright, G. Raskutti, and B. Yu. High-dimensional covariance estimation by minimizing $\ell_1$-penalized log-determinant divergence. *Electronic Journal of Statistics*, 5:935–980, 2011.

[23] A. Rothman, P. Bickel, E. Levina, and J. Zhu. Sparse permutation invariant covariance estimation. *Electronic Journal of Statistics*, 2:494–515, 2008.

[24] X. Shen, W. Pan, and Y. Zhu. ). likelihood-based selection and sharp parameter estimation. *Journal of the American Statistical Association*, 2012. to appear.

[25] R. Tibshirani. Regression shrinkage and selection via the lasso. *Journal of the Royal Statistical Society, Series B*, 58(1):267–288, 1996.

[26] D. Vogel and R. Fried. Elliptical graphical modelling. *Biometrika*, 98(4):935–951, December 2011.

[27] M. Wainwright. Sharp thresholds for highdimensional and noisy sparsity recovery using $\ell_1$ constrained quadratic programming. *IEEE Transactions on Information Theory*, 55(5):2183–2201, 2009.

[28] L. Xue and H. Zou. Regularized rank-based estimation of high-dimensional nonparanormal graphical models. *Annals of Statistics*, 2012.

[29] M. Yuan. High dimensional inverse covariance matrix estimation via linear programming. *Journal of Machine Learning Research*, 11(8):2261–2286, 2010.

[30] M. Yuan and Y. Lin. Model selection and estimation in the gaussian graphical model. *Biometrika*, 94(1):19–35, 2007.

[31] P. Zhao and B. Yu. On model selection consistency of lasso. *Journal of Machine Learning Research*, 7(11):2541–2563, 2006.

[32] T. Zhao, H. Liu, K. Roeder, J. Lafferty, and L. Wasserman. The huge package for high-dimensional undirected graph estimation in r. *Journal of Machine Learning Research*, 2012. to appear.

[33] H. Zou. The adaptive lasso and its oracle properties. *Journal of the American Statistical Association*, 101(476):1418–1429, 2006.

